# Bayesian Belief Polarization

**Alan Jern**
Department of Psychology
Carnegie Mellon University
ajern@cmu.edu

**Kai-min K. Chang**
Language Technologies Institute
Carnegie Mellon University
kkchang@cs.cmu.edu

**Charles Kemp**
Department of Psychology
Carnegie Mellon University
ckemp@cmu.edu

## Abstract

Empirical studies have documented cases of belief polarization, where two people with opposing prior beliefs both strengthen their beliefs after observing the same evidence. Belief polarization is frequently offered as evidence of human irrationality, but we demonstrate that this phenomenon is consistent with a fully Bayesian approach to belief revision. Simulation results indicate that belief polarization is not only possible but relatively common within the set of Bayesian models that we consider.

Suppose that Carol has requested a promotion at her company and has received a score of 50 on an aptitude test. Alice, one of the company's managers, began with a high opinion of Carol and became even more confident of her abilities after seeing her test score. Bob, another manager, began with a low opinion of Carol and became even less confident about her qualifications after seeing her score. On the surface, it may appear that either Alice or Bob is behaving irrationally, since the same piece of evidence has led them to update their beliefs about Carol in opposite directions. This situation is an example of belief polarization [1, 2], a widely studied phenomenon that is often taken as evidence of human irrationality [3, 4].

In some cases, however, belief polarization may appear much more sensible when all the relevant information is taken into account. Suppose, for instance, that Alice was familiar with the aptitude test and knew that it was scored out of 60, but that Bob was less familiar with the test and assumed that the score was a percentage. Even though only one interpretation of the score can be correct, Alice and Bob have both made rational inferences given their assumptions about the test.

Some instances of belief polarization are almost certain to qualify as genuine departures from rational inference, but we argue in this paper that others will be entirely compatible with a rational approach. Distinguishing between these cases requires a precise normative standard against which human inferences can be compared. We suggest that Bayesian inference provides this normative standard, and present a set of Bayesian models that includes cases where polarization can and cannot emerge. Our work is in the spirit of previous studies that use careful rational analyses in order to illuminate apparently irrational human behavior (e.g. [5, 6, 7]).

Previous studies of belief polarization have occasionally taken a Bayesian approach, but often the goal is to show how belief polarization can emerge as a consequence of approximate inference in a Bayesian model that is subject to memory constraints or processing limitations [8]. In contrast, we demonstrate that some examples of polarization are compatible with a fully Bayesian approach. Other formal accounts of belief polarization have relied on complex versions of utility theory [9], or have focused on continuous hypothesis spaces [10] unlike the discrete hypothesis spaces usually considered by psychological studies of belief polarization. We focus on discrete hypothesis spaces and require no additional machinery beyond the basics of Bayesian inference.

We begin by introducing the belief revision phenomena considered in this paper and developing a Bayesian approach that clarifies whether and when these phenomena should be considered irrational. We then consider several Bayesian models that are capable of producing belief polarization and illustrate them with concrete examples. Having demonstrated that belief polarization is compatible

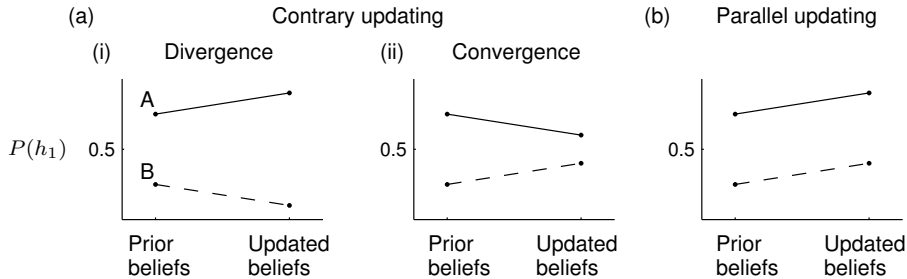

Figure 1: Examples of belief updating behaviors for two individuals, A (solid line) and B (dashed line). The individuals begin with different beliefs about hypothesis $h_1$. After observing the same set of evidence, their beliefs may (a) move in opposite directions or (b) move in the same direction.

with a Bayesian approach, we present simulations suggesting that this phenomenon is relatively generic within the space of models that we consider. We finish with some general comments on human rationality and normative models.

# 1   Belief revision phenomena

The term "belief polarization" is generally used to describe situations in which two people observe the same evidence and update their respective beliefs in the directions of their priors. A study by Lord, et al. [1] provides one classic example in which participants read about two studies, one of which concluded that the death penalty deters crime and another which concluded that the death penalty has no effect on crime. After exposure to this mixed evidence, supporters of the death penalty strengthened their support and opponents strengthened their opposition.

We will treat belief polarization as a special case of *contrary updating*, a phenomenon where two people update their beliefs in opposite directions after observing the same evidence (Figure 1a). We distinguish between two types of contrary updating. *Belief divergence* refers to cases in which the person with the stronger belief in some hypothesis increases the strength of his or her belief and the person with the weaker belief in the hypothesis decreases the strength of his or her belief (Figure 1a(i)). Divergence therefore includes cases of traditional belief polarization. The opposite of divergence is *belief convergence* (Figure 1a(ii)), in which the person with the stronger belief decreases the strength of his or her belief and the person with the weaker belief increases the strength of his or her belief. Contrary updating may be contrasted with *parallel updating* (Figure 1b), in which the two people update their beliefs in the same direction. Throughout this paper, we consider only situations in which both people change their beliefs after observing some evidence. All such situations can be unambiguously classified as instances of parallel or contrary updating.

Parallel updating is clearly compatible with a normative approach, but the normative status of divergence and convergence is less clear. Many authors argue that divergence is irrational, and many of the same authors also propose that convergence is rational [2, 3]. For example, Baron [3] writes that "Normatively, we might expect that beliefs move toward the middle of the range when people are presented with mixed evidence." (p. 210) The next section presents a formal analysis that challenges the conventional wisdom about these phenomena and clarifies the cases where they can be considered rational.

# 2   A Bayesian approach to belief revision

Since belief revision involves inference under uncertainty, Bayesian inference provides the appropriate normative standard. Consider a problem where two people observe data $d$ that bear on some hypothesis $h_1$. Let $P_1(\cdot)$ and $P_2(\cdot)$ be distributions that capture the two people's respective beliefs. Contrary updating occurs whenever one person's belief in $h_1$ increases and the other person's belief in $h_1$ decreases, or when

$$[P_1(h_1|d) - P_1(h_1)]\,[P_2(h_1|d) - P_2(h_1)] < 0\,. \tag{1}$$

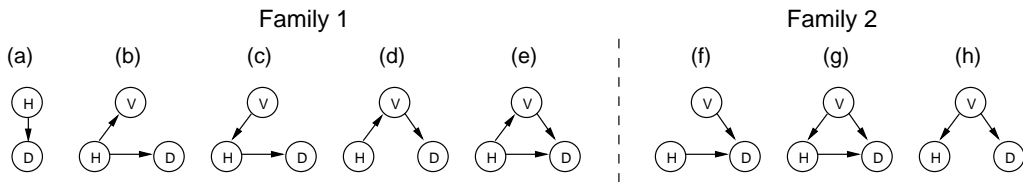

Figure 2: (a) A simple Bayesian network that cannot produce either belief divergence or belief convergence. (b) – (h) All possible three-node Bayes nets subject to the constraints described in the text. Networks in Family 1 can produce only parallel updating, but networks in Family 2 can produce both parallel and contrary updating.

We will use Bayesian networks to capture the relationships between $H$, $D$, and any other variables that are relevant to the situation under consideration. For example, Figure 2a captures the idea that the data $D$ are probabilistically generated from hypothesis $H$. The remaining networks in Figure 2 show several other ways in which $D$ and $H$ may be related, and will be discussed later.

We assume that the two individuals agree on the variables that are relevant to a problem and agree about the relationships between these variables. We can formalize this idea by requiring that both people agree on the structure and the conditional probability distributions (CPDs) of a network $N$ that captures relationships between the relevant variables, and that they differ only in the priors they assign to the root nodes of $N$. If $N$ is the Bayes net in Figure 2a, then we assume that the two people must agree on the distribution $P(D|H)$, although they may have different priors $P_1(H)$ and $P_2(H)$.

If two people agree on network $N$ but have different priors on the root nodes, we can create a single expanded Bayes net to simulate the inferences of both individuals. The expanded network is created by adding a background knowledge node $B$ that sends directed edges to all root nodes in $N$, and acts as a switch that sets different root node priors for the two different individuals. Given this expanded network, distributions $P_1$ and $P_2$ in Equation 1 can be recovered by conditioning on the value of the background knowledge node and rewritten as

$$[P(h_1|d, b_1) - P(h_1|b_1)] \, [P(h_1|d, b_2) - P(h_1|b_2)] < 0 \tag{2}$$

where $P(\cdot)$ represents the probability distribution captured by the expanded network.

Suppose that there are exactly two mutually exclusive hypotheses. For example, $h_1$ and $h_0$ might state that the death penalty does or does not deter crime. In this case Equation 2 implies that contrary updating occurs when

$$[P(d|h_1, b_1) - P(d|h_0, b_1)] \, [P(d|h_1, b_2) - P(d|h_0, b_2)] < 0 \,. \tag{3}$$

Equation 3 is derived in the supporting material, and leads immediately to the following result:

> **R1:** If $H$ is a binary variable and $D$ and $B$ are conditionally independent given $H$, then contrary updating is impossible.

Result R1 follows from the observation that if $D$ and $B$ are conditionally independent given $H$, then the product in Equation 3 is equal to $(P(d|h_1) - P(d|h_0))^2$, which cannot be less than zero.

R1 implies that the simple Bayes net in Figure 2a is incapable of producing contrary updating, an observation previously made by Lopes [11]. Our analysis may help to explain the common intuition that belief divergence is irrational, since many researchers seem to implicitly adopt a model in which $H$ and $D$ are the only relevant variables. Network 2a, however, is too simple to capture the causal relationships that are present in many real world situations. For example, the promotion example at the beginning of this paper is best captured using a network with an additional node that represents the grading scale for the aptitude test. Networks with many nodes may be needed for some real world problems, but here we explore the space of three-node networks.

We restrict our attention to connected graphs in which $D$ has no outgoing edges, motivated by the idea that the three variables should be linked and that the data are the final result of some generative process. The seven graphs that meet these conditions are shown in Figures 2b–h, where the additional variable has been labeled $V$. These Bayes nets illustrate cases in which (b) $V$ is an additional

| | Conventional wisdom | Models | |
| --- | --- | --- | --- |
| | | Family 1 | Family 2 |
| Belief divergence | | | ✓ |
| Belief convergence | ✓ | | ✓ |
| Parallel updating | ✓ | ✓ | ✓ |

Table 1: The first column represents the conventional wisdom about which belief revision phenomena are normative. The models in the remaining columns include all three-node Bayes nets. This set of models can be partitioned into those that support both belief divergence and convergence (Family 2) and those that support neither (Family 1).

piece of evidence that bears on $H$, (c) $V$ informs the prior probability of $H$, (d)–(e) $D$ is generated by an intervening variable $V$, (f) $V$ is an additional generating factor of $D$, (g) $V$ informs both the prior probability of $H$ and the likelihood of $D$, and (h) $H$ and $D$ are both effects of $V$. The graphs in Figure 2 have been organized into two families. R1 implies that none of the graphs in Family 1 is capable of producing contrary updating. The next section demonstrates by example that all three of the graphs in Family 2 are capable of producing contrary updating.

Table 1 compares the two families of Bayes nets to the informal conclusions about normative approaches that are often found in the psychological literature. As previously noted, the conventional wisdom holds that belief divergence is irrational but that convergence and parallel updating are both rational. Our analysis suggests that this position has little support. Depending on the causal structure of the problem under consideration, a rational approach should allow both divergence and convergence or neither.

Although we focus in this paper on Bayes nets with no more than three nodes, the class of all network structures can be partitioned into those that can (Family 2) and cannot (Family 1) produce contrary updating. R1 is true for Bayes nets of any size and characterizes one group of networks that belong to Family 1. Networks where the data provide no information about the hypotheses must also fail to produce contrary updating. Note that if $D$ and $H$ are conditionally independent given $B$, then the left side of Equation 3 is equal to zero, meaning contrary updating cannot occur. We conjecture that all remaining networks can produce contrary updating if the cardinalities of the nodes and the CPDs are chosen appropriately. Future studies can attempt to verify this conjecture and to precisely characterize the CPDs that lead to contrary updating.

## 3    Examples of rational belief divergence

We now present four scenarios that can be modeled by the three-node Bayes nets in Family 2. Our purpose in developing these examples is to demonstrate that these networks can produce belief divergence and to provide some everyday examples in which this behavior is both normative and intuitive.

### 3.1    Example 1: Promotion

We first consider a scenario that can be captured by Bayes net 2f, in which the data depend on two independent factors. Recall the scenario described at the beginning of this paper: Alice and Bob are responsible for deciding whether to promote Carol. For simplicity, we consider a case where the data represent a binary outcome—whether or not Carol's résumé indicates that she is included in The Directory of Notable People—rather than her score on an aptitude test. Alice believes that The Directory is a reputable publication but Bob believes it is illegitimate. This situation is represented by the Bayes net and associated CPDs in Figure 3a. In the tables, the hypothesis space $H = \{\text{'Unqualified'} = 0, \text{'Qualified'} = 1\}$ represents whether or not Carol is qualified for the promotion, the additional factor $V = \{\text{'Disreputable'} = 0, \text{'Reputable'} = 1\}$ represents whether The Directory is a reputable publication, and the data variable $D = \{\text{'Not included'} = 0, \text{'Included'} = 1\}$ represents whether Carol is featured in it. The actual probabilities were chosen to reflect the fact that only an unqualified person is likely to pad their résumé by mentioning a disreputable publication, but that

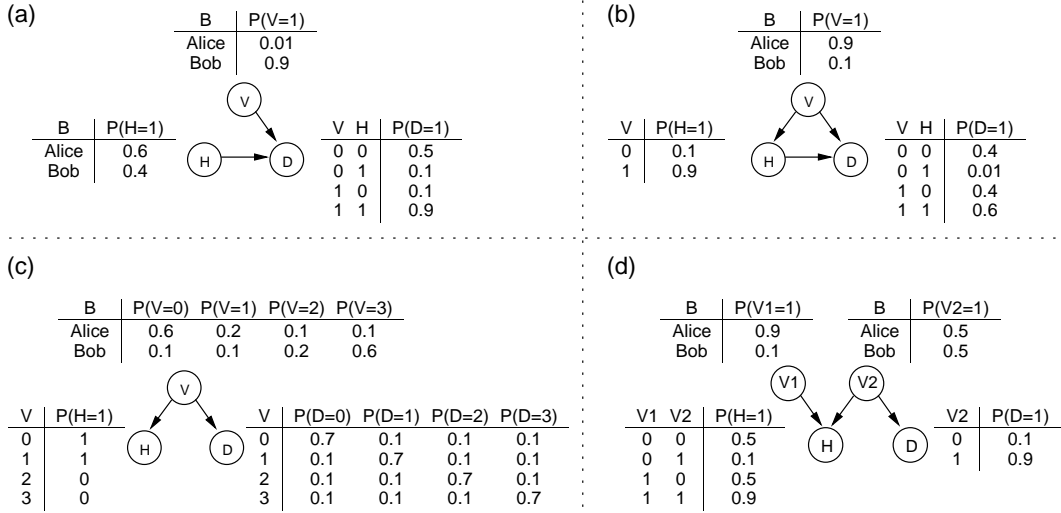

Figure 3: The Bayes nets and conditional probability distributions used in (a) Example 1: Promotion, (b) Example 2: Religious belief, (c) Example 3: Election polls, (d) Example 4: Political belief.

only a qualified person is likely to be included in The Directory if it is reputable. Note that Alice and Bob agree on the conditional probability distribution for $D$, but assign different priors to $V$ and $H$. Alice and Bob therefore interpret the meaning of Carol's presence in The Directory differently, resulting in the belief divergence shown in Figure 4a.

This scenario is one instance of a large number of belief divergence cases that can be attributed to two individuals possessing different mental models of how the observed evidence was generated. For instance, suppose now that Alice and Bob are both on an admissions committee and are evaluating a recommendation letter for an applicant. Although the letter is positive, it is not enthusiastic. Alice, who has less experience reading recommendation letters interprets the letter as a strong endorsement. Bob, however, takes the lack of enthusiasm as an indication that the author has some misgivings [12]. As in the promotion scenario, the differences in Alice's and Bob's experience can be effectively represented by the priors they assign to the $H$ and $V$ nodes in a Bayes net of the form in Figure 2f.

### 3.2 Example 2: Religious belief

We now consider a scenario captured by Bayes net 2g. In our example for Bayes net 2f, the status of an additional factor $V$ affected how Alice and Bob interpreted the data $D$, but did not shape their prior beliefs about $H$. In many cases, however, the additional factor $V$ will influence both people's prior beliefs about $H$ as well as their interpretation of the relationship between $D$ and $H$. Bayes net 2g captures this situation, and we provide a concrete example inspired by an experiment conducted by Batson [13].

Suppose that Alice believes in a "Christian universe:" she believes in the divinity of Jesus Christ and expects that followers of Christ will be persecuted. Bob, on the other hand, believes in a "secular universe." This belief leads him to doubt Christ's divinity, but to believe that if Christ were divine, his followers would likely be protected rather than persecuted. Now suppose that both Alice and Bob observe that Christians are, in fact, persecuted, and reassess the probability of Christ's divinity. This situation is represented by the Bayes net and associated CPDs in Figure 3b. In the tables, the hypothesis space $H = \{\text{'Human'} = 0, \text{'Divine'} = 1\}$ represents the divinity of Jesus Christ, the additional factor $V = \{\text{'Secular'} = 0, \text{'Christian'} = 1\}$ represents the nature of the universe, and the data variable $D = \{\text{'Not persecuted'} = 0, \text{'Persecuted'} = 1\}$ represents whether Christians are subject to persecution. The exact probabilities were chosen to reflect the fact that, regardless of worldview, people will agree on a "base rate" of persecution given that Christ is not divine, but that more persecution is expected if the Christian worldview is correct than if the secular worldview is correct. Unlike in the previous scenario, Alice and Bob agree on the CPDs for both $D$ and $H$, but

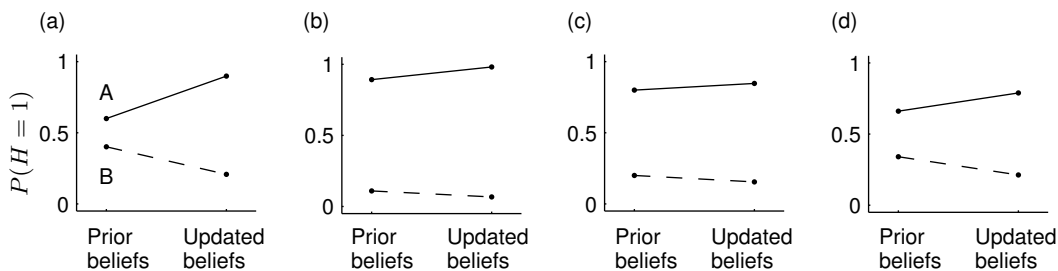

Figure 4: Belief revision outcomes for (a) Example 1: Promotion, (b) Example 2: Religious belief, (c) Example 3: Election polls, and (d) Example 4: Political belief. In all four plots, the updated beliefs for Alice (solid line) and Bob (dashed line) are computed after observing the data described in the text. The plots confirm that all four of our example networks can lead to belief divergence.

differ in the priors they assign to $V$. As a result, Alice and Bob disagree about whether persecution supports or undermines a Christian worldview, which leads to the divergence shown in Figure 4b.

This scenario is analogous to many real world situations in which one person has knowledge that the other does not. For instance, in a police interrogation, someone with little knowledge of the case ($V$) might take a suspect's alibi ($D$) as strong evidence of their innocence ($H$). However, a detective with detailed knowledge of the case may assign a higher prior probability to the subject's guilt based on other circumstantial evidence, and may also notice a detail in the suspect's alibi that only the culprit would know, thus making the statement strong evidence of guilt. In all situations of this kind, although two people possess different background knowledge, their inferences are normative given that knowledge, consistent with the Bayes net in Figure 2g.

### 3.3   Example 3: Election polls

We now consider two qualitatively different cases that are both captured by Bayes net 2h. The networks considered so far have all included a direct link between $H$ and $D$. In our next two examples, we consider cases where the hypotheses and observed data are not directly linked, but are coupled by means of one or more unobserved causal factors.

Suppose that an upcoming election will be contested by two Republican candidates, Rogers and Rudolph, and two Democratic candidates, Davis and Daly. Alice and Bob disagree about the various candidates' chances of winning, with Alice favoring the two Republicans and Bob favoring the two Democrats. Two polls were recently released, one indicating that Rogers was most likely to win the election and the other indicating that Daly was most likely to win. After considering these polls, they both assess the likelihood that a Republican will win the election.

This situation is represented by the Bayes net and associated CPDs in Figure 3c. In the tables, the hypothesis space $H = \{$'Democrat wins' $= 0$, 'Republican wins' $= 1\}$ represents the winning party, the variable $V = \{$'Rogers' $= 0$, 'Rudolph' $= 1$, 'Davis' $= 2$, 'Daly' $= 3\}$ represents the winning candidate, and the data variables $D1 = D2 = \{$'Rogers' $= 0$, 'Rudolph' $= 1$, 'Davis' $= 2$, 'Daly' $= 3\}$ represent the results of the two polls. The exact probabilities were chosen to reflect the fact that the polls are likely to reflect the truth with some noise, but whether a Democrat or Republican wins is completely determined by the winning candidate $V$. In Figure 3c, only a single $D$ node is shown because $D1$ and $D2$ have identical CPDs. The resulting belief divergence is shown in Figure 4c.

Note that in this scenario, Alice's and Bob's different priors cause them to discount the poll that disagrees with their existing beliefs as noise, thus causing their prior beliefs to be reinforced by the mixed data. This scenario was inspired by the death penalty study [1] alluded to earlier, in which a set of mixed results caused supporters and opponents of the death penalty to strengthen their existing beliefs. We do not claim that people's behavior in this study can be explained with exactly the model employed here, but our analysis does show that selective interpretation of evidence is sometimes consistent with a rational approach.

### 3.4 Example 4: Political belief

We conclude with a second illustration of Bayes net 2h in which two people agree on the interpretation of an observed piece of evidence but disagree about the implications of that evidence. In this scenario, Alice and Bob are two economists with different philosophies about how the federal government should approach a major recession. Alice believes that the federal government should increase its own spending to stimulate economic activity; Bob believes that the government should decrease its spending and reduce taxes instead, providing taxpayers with more spending money. A new bill has just been proposed and an independent study found that the bill was likely to increase federal spending. Alice and Bob now assess the likelihood that this piece of legislation will improve the economic climate.

This scenario can be modeled by the Bayes net and associated CPDs in Figure 3d. In the tables, the hypothesis space $H = \{$'Bad policy' $= 0$, 'Good policy' $= 1\}$ represents whether the new bill is good for the economy and the data variable $D = \{$'No spending' $= 0$, 'Spending increase' $= 1\}$ represents the conclusions of the independent study. Unlike in previous scenarios, we introduce two additional factors, $V1 = \{$'Fiscally conservative' $= 0$, 'Fiscally liberal' $= 1\}$, which represents the optimal economic philosophy, and $V2 = \{$'No spending' $= 0$, 'Spending increase' $= 1\}$, which represents the spending policy of the new bill. The exact probabilities in the tables were chosen to reflect the fact that if the bill does not increase spending, the policy it enacts may still be good for other reasons. A uniform prior was placed on $V2$ for both people, reflecting the fact that they have no prior expectations about the spending in the bill. However, the priors placed on $V1$ for Alice and Bob reflect their different beliefs about the best economic policy. The resulting belief divergence behavior is shown in Figure 4d. The model used in this scenario bears a strong resemblance to the probabilogical model of attitude change developed by McGuire [14] in which $V1$ and $V2$ might be logical "premises" that entail the "conclusion" $H$.

## 4  How common is contrary updating?

We have now described four concrete cases where belief divergence is captured by a normative approach. It is possible, however, that belief divergence is relatively rare within the Bayes nets of Family 2, and that our four examples are exotic special cases that depend on carefully selected CPDs. To rule out this possibility, we ran simulations to explore the space of all possible CPDs for the three networks in Family 2.

We initially considered cases where $H$, $D$, and $V$ were binary variables, and ran two simulations for each model. In one simulation, the priors and each row of each CPD were sampled from a symmetric Beta distribution with parameter $0.1$, resulting in probabilities highly biased toward $0$ and $1$. In the second simulation, the probabilities were sampled from a uniform distribution. In each trial, a single set of CPDs were generated and then two different priors were generated for each root node in the graph to simulate two individuals, consistent with our assumption that two individuals may have different priors but must agree about the conditional probabilities. 20,000 trials were carried out in each simulation, and the proportion of trials that led to convergence and divergence was computed. Trials were only counted as instances of convergence or divergence if $|P(H = 1|D = 1) - P(H = 1)| > \epsilon$ for both individuals, with $\epsilon = 1 \times 10^{-5}$.

The results of these simulations are shown in Table 2. The supporting material proves that divergence and convergence are equally common, and therefore the percentages in the table show the frequencies for contrary updating of either type. Our primary question was whether contrary updating is rare or anomalous. In all but the third simulation, contrary updating constituted a substantial proportion of trials, suggesting that the phenomenon is relatively generic. We were also interested in whether this behavior relied on particular settings of the CPDs. The fact that percentages for the uniform distribution are approximately the same or greater than for the biased distribution indicates that contrary updating appears to be a relatively generic behavior for the Bayes nets we considered. More generally, these results directly challenge the suggestion that normative accounts are not suited for modeling belief divergence.

The last two columns of Table 2 show results for two simulations with the same Bayes net, the only difference being whether $V$ was treated as 2-valued (binary) or 4-valued. The 4-valued case is included because both Examples 3 and 4 considered multi-valued additional factor variables $V$.

|  | 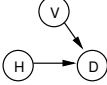 | | 2-valued $V$ 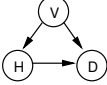 | 4-valued $V$ |
|---|---|---|---|---|
| Biased | 9.6% | 12.7% | 0% | 23.3% |
| Uniform | 18.2% | 16.0% | 0% | 20.0% |

Table 2: Simulation results. The percentages indicate the proportion of trials that produced contrary updating using the specified Bayes net (column) and probability distributions (row). The prior and conditional probabilities were either sampled from a Beta$(0.1, 0.1)$ distribution (biased) or a Beta$(1, 1)$ distribution (uniform). The probabilities for the simulation results shown in the last column were sampled from a Dirichlet$([0.1, 0.1, 0.1, 0.1])$ distribution (biased) or a Dirichlet$([1, 1, 1, 1])$ distribution (uniform).

In Example 4, we used two binary variables, but we could have equivalently used a single 4-valued variable. Belief convergence and divergence are not possible in the binary case, a result that is proved in the supporting material. We believe, however, that convergence and divergence are fairly common whenever $V$ takes three or more values, and the simulation in the last column of the table confirms this claim for the 4-valued case.

Given that belief divergence seems relatively common in the space of all Bayes nets, it is natural to explore whether cases of rational divergence are regularly encountered in the real world. One possible approach is to analyze a large database of networks that capture everyday belief revision problems, and to determine what proportion of networks lead to rational divergence. Future studies can explore this issue, but our simulations suggest that contrary updating is likely to arise in cases where it is necessary to move beyond a simple model like the one in Figure 2a and consider several causal factors.

## 5   Conclusion

This paper presented a family of Bayes nets that can account for belief divergence, a phenomenon that is typically considered to be incompatible with normative accounts. We provided four concrete examples that illustrate how this family of networks can capture a variety of settings where belief divergence can emerge from rational statistical inference. We also described a series of simulations that suggest that belief divergence is not only possible but relatively common within the family of networks that we considered.

Our work suggests that belief polarization should not always be taken as evidence of irrationality, and that researchers who aim to document departures from rationality may wish to consider alternative phenomena instead. One such phenomenon might be called "inevitable belief reinforcement" and occurs when supporters of a hypothesis update their belief in the same direction for all possible data sets $d$. For example, a gambler will demonstrate inevitable belief reinforcement if he or she becomes increasingly convinced that a roulette wheel is biased towards red regardless of whether the next spin produces red, black, or green. This phenomenon is provably inconsistent with any fully Bayesian approach, and therefore provides strong evidence of irrationality.

Although we propose that some instances of polarization are compatible with a Bayesian approach, we do not claim that human inferences are always or even mostly rational. We suggest, however, that characterizing normative behavior can require careful thought, and that formal analyses are invaluable for assessing the rationality of human inferences. In some cases, a formal analysis will provide an appropriate baseline for understanding how human inferences depart from rational norms. In other cases, a formal analysis will suggest that an apparently irrational inference makes sense once all of the relevant information is taken into account.

# References

[1] C. G. Lord, L. Ross, and M. R. Lepper. Biased assimilation and attitude polarization: The effects of prior theories on subsequently considered evidence. *Journal of Personality and Social Psychology*, 37(1):2098–2109, 1979.

[2] L. Ross and M. R. Lepper. The perseverance of beliefs: Empirical and normative considerations. In *New directions for methodology of social and behavioral science: Fallible judgment in behavioral research*. Jossey-Bass, San Francisco, 1980.

[3] J. Baron. *Thinking and Deciding*. Cambridge University Press, Cambridge, 4th edition, 2008.

[4] A. Gerber and D. Green. Misperceptions about perceptual bias. *Annual Review of Political Science*, 2:189–210, 1999.

[5] M. Oaksford and N. Chater. A rational analysis of the selection task as optimal data selection. *Psychological Review*, 101(4):608–631, 1994.

[6] U. Hahn and M. Oaksford. The rationality of informal argumentation: A Bayesian approach to reasoning fallacies. *Psychological Review*, 114(3):704–732, 2007.

[7] S. Sher and C. R. M. McKenzie. Framing effects and rationality. In N. Chater and M. Oaksford, editors, *The probablistic mind: Prospects for Bayesian cognitive science*. Oxford University Press, Oxford, 2008.

[8] B. O'Connor. Biased evidence assimilation under bounded Bayesian rationality. Master's thesis, Stanford University, 2006.

[9] A. Zimper and A. Ludwig. Attitude polarization. Technical report, Mannheim Research Institute for the Economics of Aging, 2007.

[10] A. K. Dixit and J. W. Weibull. Political polarization. *Proceedings of the National Academy of Sciences*, 104(18):7351–7356, 2007.

[11] L. L. Lopes. Averaging rules and adjustment processes in Bayesian inference. *Bulletin of the Psychonomic Society*, 23(6):509–512, 1985.

[12] A. Harris, A. Corner, and U. Hahn. "Damned by faint praise": A Bayesian account. In A. D. De Groot and G. Heymans, editors, *Proceedings of the 31th Annual Conference of the Cognitive Science Society*, Austin, TX, 2009. Cognitive Science Society.

[13] C. D. Batson. Rational processing or rationalization? The effect of disconfirming information on a stated religious belief. *Journal of Personality and Social Psychology*, 32(1):176–184, 1975.

[14] W. J. McGuire. The probabilogical model of cognitive structure and attitude change. In R. E. Petty, T. M. Ostrom, and T. C. Brock, editors, *Cognitive Responses in Persuasion*. Lawrence Erlbaum Associates, 1981.

